# Building Predictive Models from Fractal Representations of Symbolic Sequences

**Peter Tiňo   Georg Dorffner**
Austrian Research Institute for Artificial Intelligence
Schottengasse 3, A-1010 Vienna, Austria
{*petert,georg*}*@ai.univie.ac.at*

## Abstract

We propose a novel approach for building finite memory predictive models similar in spirit to variable memory length Markov models (VLMMs). The models are constructed by first transforming the $n$-block structure of the training sequence into a spatial structure of points in a unit hypercube, such that the longer is the common suffix shared by any two $n$-blocks, the closer lie their point representations. Such a transformation embodies a Markov assumption – $n$-blocks with long common suffixes are likely to produce similar continuations. Finding a set of prediction contexts is formulated as a resource allocation problem solved by vector quantizing the spatial $n$-block representation. We compare our model with both the classical and variable memory length Markov models on three data sets with different memory and stochastic components. Our models have a superior performance, yet, their construction is fully automatic, which is shown to be problematic in the case of VLMMs.

## 1   Introduction

Statistical modeling of complex sequences is a prominent theme in machine learning due to its wide variety of applications (see e.g. [5]). Classical Markov models (MMs) of finite order are simple, yet widely used models for sequences generated by stationary sources. However, MMs can become hard to estimate due to the familiar explosive increase in the number of free parameters when increasing the model order. Consequently, only low order MMs can be considered in practical applications. Some time ago, Ron, Singer and Tishby [4] introduced at this conference a Markovian model that could (at least partially) overcome the curse of dimensionality in classical MMs. The basic idea behind their model was simple: instead of fixed-order MMs consider variable memory length Markov models (VLMMs) with a "deep" memory just where it is really needed (see also e.g. [5][7]).

The size of VLMMs is usually controlled by one or two construction parameters. Unfortunately, constructing a series of increasingly complex VLMMs (for example to enter a model selection phase on a validation set) by varying the construction parameters can be

a troublesome task [1]. Construction often does not work "smoothly" with varying the parameters. There are large intervals of parameter values yielding unchanged VLMMs interleaved with tiny parameter regions corresponding to a large spectrum of VLMM sizes. In such cases it is difficult to fully automize the VLMM construction.

To overcome this drawback, we suggest an alternative predictive model similar in spirit to VLMMs. Searching for the relevant prediction contexts is reformulated as a resource allocation problem in Euclidean space solved by vector quantization. A potentially prohibitively large set of all length-$L$ blocks is assigned to a much smaller set of prediction contexts on a suffix basis. To that end, we first transform the set of $L$-blocks appearing in the training sequence into a set of points in Euclidean space, such that points corresponding to blocks sharing a long common suffix are mapped close to each other. Vector quantization on such a set partitions the set of $L$-blocks into several classes dominated by common suffixes. Quantization centers play the role of predictive contexts. A great advantage of our model is that vector quantization can be performed on a completely self-organized basis.

We compare our model with both classical MMs and VLMMs on three data sets representing a wide range of grammatical and statistical structure. First, we train the models on the Feigenbaum binary sequence with a very strict topological and metric organization of allowed subsequences. Highly specialized, deep prediction contexts are needed to model this sequence. Classical Markov models cannot succeed and the full power of admitting a limited number of variable length contexts can be exploited. The second data set consists of quantized daily volatility changes of the Dow Jones Industrial Average (DJIA). Predictive models are used to predict the direction of volatility move for the next day. Financial time series are known to be highly stochastic with a relatively shallow memory structure. In this case, it is difficult to beat low-order classical MMs. One can perform better than MMs only by developing a few deeper specialized contexts, but that, on the other hand, can lead to overfitting. Finally, we test our model on the experiments of Ron, Singer and Tishby with language data from the Bible [5]. They trained classical MMs and a VLMM on the books of the Bible except for the book of Genesis. Then the models were evaluated on the bases of negative log-likelihood on an unseen text from Genesis. We compare likelihood results of our model with those of MMs and VLMMs.

## 2   Predictive models

We consider sequences $S = s_1 s_2 ...$ over a finite alphabet $\mathcal{A} = \{1, 2, ..., A\}$ generated by stationary sources. The set of all sequences over $\mathcal{A}$ with exactly $n$ symbols is denoted by $\mathcal{A}^n$.

An information source over $\mathcal{A} = \{1, 2, ..., A\}$ is defined by a family of consistent probability measures $P_n$ on $\mathcal{A}^n$, $n = 0, 1, 2, ...$, $\sum_{s \in \mathcal{A}} P_{n+1}(ws) = P_n(w)$, for all $w \in \mathcal{A}^n$ ($\mathcal{A}^0 = \{\Lambda\}$ and $P_0(\Lambda) = 1$, $\Lambda$ denotes the empty string).

In applications it is useful to consider probability functions $P_n$ that are easy to handle. This can be achieved, for example, by assuming a finite source memory of length at most $L$, and formulating the conditional measures $P(s|w) = P_{L+1}(ws)/P_L(w)$, $w \in \mathcal{A}^L$, using a function $c : \mathcal{A}^L \to \mathcal{C}$, from $L$-blocks over $\mathcal{A}$ to a (presumably small) finite set $\mathcal{C}$ of prediction contexts:

$$P(s|w) = P(s|c(w)). \tag{1}$$

In *Markov models* (MMs) of order $n \leq L$, for all $L$-blocks $w \in \mathcal{A}^L$, $c(w)$ is the length-$n$

suffix of $w$, i.e. $c(uv) = v$, $v \in \mathcal{A}^n$, $u \in \mathcal{A}^{L-n}$.

In *variable memory length Markov models* (VLMMs), the suffices $c(w)$ of $L$-blocks $w \in \mathcal{A}^L$ can have different lengths, depending on the particular $L$-block $w$. For strategies of selecting and representing the prediction contexts through prediction suffix trees and/or probabilistic suffix automata see, for example, [4][5]. VLMM construction is controlled by one, or several parameters regulating selection of candidate contexts and growing/pruning decisions.

Prediction context function $c : \mathcal{A}^L \to \mathcal{C}$ in Markov models of order $n \leq L$, can be interpreted as a natural homomorphism $c : \mathcal{A}^L \to \mathcal{A}^L|_{\mathcal{E}}$ corresponding to the equivalence relation $\mathcal{E} \subseteq \mathcal{A}^L \times \mathcal{A}^L$ on $L$-blocks over $\mathcal{A}$: two $L$-blocks $u, v$ are in the same class, i.e. $(u, v) \in \mathcal{E}$, if they share the same suffix of length $n$. The factor set $\mathcal{A}^L|_{\mathcal{E}} = \mathcal{C} = \mathcal{A}^n$ consists of all $n$-blocks over $\mathcal{A}$. Classical MMs define the equivalence $\mathcal{E}$ on the suffix bases, but *regardless of the suffix structure present in the training data*. Our idea is to keep the Markov-motivated suffix strategy for constructing $\mathcal{E}$, but at the same time *take into an account the data suffix structure*.

Vector quantization on a set of $B$ points in a Euclidean space positions $N << B$ codebook vectors (CVs), each CV representing a subset of points that are closer to it than to any other CV, so that the overall error of substituting CVs for points they represent is minimal. In other words, CVs tend to represent points lying close to each other (in a Euclidean metric). In order to use vector quantization for determining relevant predictive contexts we need to do two things:

1. Define a suitable metric in the sequence space that would correspond to Markov assumptions:

   (a) two sequences are "close" if they share a common suffix

   (b) the longer is the common suffix the closer are the sequences

2. Define a uniformly continuous map from the sequence metric space to the Euclidean space, i.e. sequences that are close in the sequence space (i.e. share a long common suffix) are mapped close to each other in the Euclidean space.

In [6] we rigorously study a class of such spatial representations of symbolic structures. Specifically, a family of distances between two $L$-blocks $u = u_1 u_2 ... u_{L-1} u_L$ and $v = v_1 v_2 ... v_{L-1} v_L$ over $\mathcal{A} = \{1, 2, ..., A\}$, expressed as

$$d_k(u, v) = \sum_{i=1}^{L} k^{L-i+1} \delta(u_i, v_i), \quad k \leq \frac{1}{2}, \tag{2}$$

with $\delta(i, j) = 1$ if $i = j$, and $\delta(i, j) = 0$ otherwise, correspond to Markov assumption. The parameter $k$ influences the rate of "forgetting the past". We construct a map from the sequence metric space to the Euclidean space as follows: Associate with each symbol $i \in \mathcal{A}$ a map

$$i(x) = kx + (1 - k)t_i, \quad t_i \in \{0, 1\}^D, \quad x \in [0, 1]^D \tag{3}$$

operating on a unit $D$-dimensional hypercube $[0, 1]^D$. Dimension of the hypercube should be large enough so that each symbol $i$ is associated with a unique vertex, i.e. $D = \lceil \log_2 A \rceil$ and $t_i \neq t_j$ whenever $i \neq j$. The map $\sigma : \mathcal{A}^L \to [0, 1]^D$, from $L$-blocks $v_1 v_2 ... v_L$ over $\mathcal{A}$ to the unit hypercube,

$$\sigma(v_1 v_2 ... v_L) = v_L(v_{L-1}(...(v_2(v_1(x^*)))...)) = (v_L \circ v_{L-1} \circ ... \circ v_2 \circ v_1)(x^*), \tag{4}$$

where $x^* = \{\frac{1}{2}\}^D$ is the center of the hypercube, is "uniformly continuous". Indeed, whenever two sequences $u, v$ share a common suffix of length $Q$, the Euclidean distance between their point representations $\sigma(u)$ and $\sigma(v)$ is less than $\sqrt{2}k^Q$. Strictly speaking, for a mathematically correct treatment of uniform continuity, we would need to consider infinite sequences. Finite blocks of symbols would then correspond to cylinder sets (see [6]). For sake of simplicity we only deal with finite sequences.

As with classical Markov models, we define the prediction context function $c : \mathcal{A}^L \to \mathcal{C}$ via an equivalence $\mathcal{E}$ on $L$-blocks over $\mathcal{A}$: two $L$-blocks $u, v$ are in the same class if their images under the map $\sigma$ are represented by the same codebook vector. In this case, the set of prediction contexts $\mathcal{C}$ can be identified with the set of codebook vectors $\{b_1, b_2, ..., b_N\}$, $b_i \in \Re^D$, $i = 1, 2, ..., N$. We refer to predictive models with such a context function as *prediction fractal machines* (PFMs). The prediction probabilities (1) are determined by

$$P(s|b_i) = \frac{N(i, s)}{\sum_{a \in \mathcal{A}} N(i, a)}, \quad s \in \mathcal{A}, \tag{5}$$

where $N(i, a)$ is the number of $(L+1)$-blocks $ua$, $a \in \mathcal{A}^L$, $a \in \mathcal{A}$, in the training sequence, such that the point $\sigma(u)$ is allocated to the codebook vector $b_i$.

## 3 Experiments

In all experiments we constructed PFMs using a contraction coefficient $k = \frac{1}{2}$ (see eq. (3)) and K-means as a vector quantization tool.

The first data set is the Feigenbaum sequence over the binary alphabet $\mathcal{A} = \{1, 2\}$. This sequence is well-studied in symbolic dynamics and has a number of interesting properties. First, the topological structure of the sequence can only be described using a context sensitive tool – a restricted indexed context-free grammar. Second, for each block length $n = 1, 2, ...,$ the distribution of $n$-blocks is either uniform, or has just two probability levels. Third, the $n$-block distributions are organized in a self-similar fashion (see [2]). The sequence can be specified by the subsequence composition rule

$$a_0 = 2, \quad a_1 = 21, \quad a_{n+1} = a_n a_{n-1} a_{n-1}. \tag{6}$$

We chose to work with the Feigenbaum sequence, because increasingly accurate modeling of the sequence with finite memory models requires a selective mechanism for deep prediction contexts.

We created a large portion of the Feigenbaum sequence and trained a series of classical MMs, variable memory length MMs (VLMMs), and prediction fractal machines (PFMs) on the first 260,000 symbols. The following 200,000 symbols formed a test set. Maximum memory length $L$ for VLMMs and PFMs was set to 30.

As mentioned in the introduction, constructing a series of increasingly complex VLMMs by varying the construction parameters appeared to be a troublesome task. We spent a fair amount of time finding "critical" parameter values at which the model size changed. In contrast, a fully automatic construction of PFMs involved sliding a window of length $L = 30$ through the training set; for each window position, mapping the $L$-block $u$ appearing in the window to the point $\sigma(u)$ (eq. (4)), vector-quantizing the resulting set of points (up to 30 codebook vectors). After the quantization step we computed predictive probabilities according to eq. (5).

Table 1: Normalized negative log-likelihoods (NNL) on the Feigenbaum test set.

| model | # contexts | NNL | captured block distribution |
|-------|-----------|------|---------------------------|
| PFM   | 2–4       | 0.6666 | 1–3 |
|       | 5–7       | 0.3333 | 1–6 |
|       | 8–22      | 0.1666 | 1–12 |
|       | 23–       | 0.0833 | 1–24 |
| VLMM  | 2–4       | 0.6666 | 1–3 |
|       | 5         | 0.3333 | 1–6 |
|       | 11        | 0.1666 | 1–12 |
|       | 23        | 0.0833 | 1–24 |
| MM    | 2,4,8,16,32 | 0.6666 | 1–3 |

Negative log-likelihoods per symbol (the base of logarithm is always taken to be the number of symbols in the alphabet) of the test set computed using the fitted models exhibited a step-like increasing tendency shown in Table 1. We also investigated the ability of the models to reproduce the $n$-block distribution found in the training and test sets. This was done by letting the models generate sequences of length equal to the length of the training sequence and for each block length $n = 1, 2, ..., 30$, computing the $L_1$ distance between the $n$-block distribution of the training and model-generated sequences. The $n$-block distributions on the test and training sets were virtually the same for $n = 1, 2, ...30$. In Table 1 we show block lengths for which the $L_1$ distance does not exceed a small threshold $\Delta$. We set $\Delta = 0.005$, since in this experiment, either the $L_1$ distance was less 0.005, or exceeded 0.005 by a large amount.

An explanation of the step-like behavior in the log-likelihood and $n$-block modeling behavior of VLMMs and PFMs is out of the scope of this paper. We briefly mention, however, that by combining the knowledge about the topological and metric structures of the Feigenbaum sequence (e.g. [2]) with a careful analysis of the models, one can show why and when an inclusion of a prediction context leads to an abrupt improvement in the modeling performance. In fact, we can show that VLMMs and PFMs constitute increasingly better approximations to the infinite self-similar Feigenbaum machine known in symbolic dynamics [2].

The classical MM totally fails in this experiment, since the context length 5 is far too small to enable the MM to mimic the complicated subsequence structure in the Feigenbaum sequence. PFMs and VLMMs quickly learn to explore a limited number of deep prediction contexts and perform comparatively well.

In the second experiment, a time series $\{x_t\}$ of the daily values of the Dow Jones Industrial Average (DJIA) from Feb. 1 1918 until April 1 1997 was transformed into a time series of returns $r_t = \log x_{t+1} - \log x_t$, and divided into 12 partially overlapping epochs, each containing about 2300 values (spanning approximately 9 years). We consider the squared return $r_t^2$ a volatility estimate for day $t$. Volatility change forecasts (volatility is going to increase or decrease) based on historical returns can be interpreted as a buying or selling signal for a straddle (see e.g. [3]). If the volatility decreases we go short (straddle is sold), if it increases we take a long position (straddle is bought). In this respect, the quality of a volatility model can be measured by the percentage of correctly predicted directions of daily volatility differences.

Table 2: Prediction performance on the DJIA volatility series.

| model | Percent correct on test set | | | | | |
|---|---|---|---|---|---|---|
|  | 1 | 2 | 3 | 4 | 5 | 6 |
| PFM | 71.08 | 70.39 | 69.70 | 70.05 | 72.12 | 72.46 |
| VLMM | 68.67 | 68.18 | 68.79 | 69.25 | 69.41 | 68.29 |
| MM | 68.56 | 69.11 | 69.78 | 68.28 | 69.50 | 73.13 |
|  | 7 | 8 | 9 | 10 | 11 | 12 |
| PFM | 74.01 | 71.77 | 73.84 | 73.84 | 71,77 | 74.19 |
| VLMM | 69.83 | 67.00 | 67.96 | 70.76 | 69.80 | 70.25 |
| MM | 74.16 | 71.96 | 69.95 | 69.16 | 71.74 | 71.07 |

The series $\{r_{t+1}^2 - r_t^2\}$ of differences between the successive squared returns is transformed into a sequence $\{D_t\}$ over 4 symbols by quantizing the series $\{r_{t+1}^2 - r_t^2\}$ as follows:

$$D_t = \begin{cases} 1 \text{ (extreme down)}, & \text{if } r_{t+1}^2 - r_t^2 < \theta_1 < 0 \\ 2 \text{ (normal down)}, & \text{if } \theta_1 \leq r_{t+1}^2 - r_t^2 < 0 \\ 3 \text{ (normal up)}, & \text{if } 0 \leq r_{t+1}^2 - r_t^2 < \theta_2 \\ 4 \text{ (extreme up)}, & \text{if } \theta_2 \leq r_{t+1}^2 - r_t^2, \end{cases} \quad (7)$$

where the parameters $\theta_1$ and $\theta_2$ correspond to $Q$ percent and $(100 - Q)$ percent sample quantiles, respectively. So, the upper (lower) $Q\%$ of all daily volatility increases (decreases) in the sample are considered extremal, and the lower (upper) $(50 - Q)\%$ of daily volatility increases (decreases) are viewed as normal.

Each epoch is partitioned into training, validation and test parts containing 110, 600 and 600 symbols, respectively. Maximum memory length $L$ for VLMMs and PFMs was set to 10 (two weeks). We trained classical MMs, VLMMs and PFMs with various numbers of prediction contexts (up to 256) and extremal event quantiles $Q \in \{5, 10, 15, ..., 45\}$. For each model class, the model size and the quantile $Q$ to be used on the test set were selected according to the validation set performance. Performance of the models was quantified as the percentage of correct guesses of the volatility change direction for the next day. If the next symbol is 1 or 2 (3 or 4) and the sum of conditional next symbol probabilities for 1 and 2 (3 and 4) given by a model is greater than 0.5, the model guess is considered correct. Results are shown in Table 2. Paired t-test reveals that PFMs significantly ($p < 0.005$) outperform both VLMMs and classical MMs.

Of course, fixed-order MMs are just special cases of VLMMs, so theoretically, VLMMs cannot perform worse than MMs. We present separate results for MMs and VLMMs to illustrate *practical* problems in fitting VLMMs. Besides familiar problems with setting the construction parameter values, one-parameter-schemes (like that presented in [4] and used here) operate only on small subsets of potential VLMMs. On data sets with a rather shallow memory structure, this can have a negative effect.

The third experiment extends the work of Ron, Singer and Tishby [5]. They tested classical MMs and VLMMs on the Bible. The alphabet is English letters and the blank character (27 symbols). The training set consisted of the Bible except for the book of Genesis. The test set was a portion of 236 characters from the book of Genesis. They set the maximal memory depth to $L = 30$ and constructed a VLMM with about 3000 contexts. Summarizing the results in [5], classical MMs of order 0, 1, 2 and 3 achieved negative log-likelihoods per

character (NNL) of 0.853, 0.681, 0.560 and 0.555, respectively. The authors point out a huge difference between the number of states in MMs of order 2 and 3: $27^3 - 27^2 = 18954$. **VLMM** performed much better and achieved an NNL of **0.456**. In our experiments, we set the maximal memory length to $L = 30$ (the same maximal memory length was used for VLMM construction in [5]). PFMs were constructed by vector quantizing a 5-dimensional (alphabet has 27 symbols) spatial representation of 30-blocks appearing in the training set. On the test set, **PFMs** with 100, 500, 1000 and 3000 predictive contexts achieved an NNL of 0.622, 0.518, 0.510 and **0.435**.

## 4 Conclusion

We presented a novel approach for building finite memory predictive models similar in spirit to variable memory length Markov models (VLMMs). Constructing a series of VLMMs is often a troublesome and highly time-consuming task requiring a lot of interactive steps. Our predictive models, prediction fractal machines (PFMs), can be constructed in a completely automatic and intuitive way - the number of codebook vectors in the vector quantization PFM construction step corresponds to the number of predictive contexts.

We tested our model on three data sets with different memory and stochastic components. VLMMs excel over the classical MMs on the Feigenbaum sequence requiring deep prediction contexts. On this sequence, PFMs achieved the same performance as their rivals - VLMMs. On financial time series, PFMs significantly outperform the purely symbolic Markov models - MMs and VLMMs. On natural language Bible data, our PFM outperforms a VLMM of comparable size.

### Acknowledgments

This work was supported by the Austrian Science Fund (FWF) within the research project "Adaptive Information Systems and Modeling in Economics and Management Science" (SFB 010) and the Slovak Academy of Sciences grant SAV 2/6018/99. The Austrian Research Institute for Artificial Intelligence is supported by the Austrian Federal Ministry of Science and Transport.

### References

[1] P. Bühlmann. Model selection for variable length Markov chains and tuning the context algorithm. *Annals of the Institute of Statistical Mathematics*, (in press), 1999.

[2] J. Freund, W. Ebeling, and K. Rateitschak. Self-similar sequences and universal scaling of dynamical entropies. *Physical Review E*, 54(5), pp. 5561–5566, 1996.

[3] J. Noh, R.F. Engle, and A. Kane. Forecasting volatility and option prices of the s&p 500 index. *Journal of Derivatives*, pp. 17–30, 1994.

[4] D. Ron, Y. Singer, and N. Tishby. The power of amnesia. In *Advances in Neural Information Processing Systems 6*, pp. 176–183. Morgan Kaufmann, 1994.

[5] D. Ron, Y. Singer, and N. Tishby. The power of amnesia. *Machine Learning*, 25, 1996.

[6] P. Tiňo. Spatial representation of symbolic sequences through iterative function system. *IEEE Transactions on Systems, Man, and Cybernetics Part A: Systems and Humans*, 29(4), pp. 386–392, 1999.

[7] M.J. Weinberger, J.J. Rissanen, and M. Feder. A universal finite memory source. *IEEE Transactions on Information Theory*, 41(3), pp. 643–652, 1995.
